# Experience-Guided Search:
# A Theory of Attentional Control

**Michael C. Mozer**
Department of Computer Science and
Institute of Cognitive Science
University of Colorado
`mozer@colorado.edu`

**David Baldwin**
Department of Computer Science
Indiana University
Bloomington, IN 47405
`baldwind@indiana.edu`

## Abstract

People perform a remarkable range of tasks that require search of the visual environment for a target item among distractors. The Guided Search model (Wolfe, 1994, 2007), or GS, is perhaps the best developed psychological account of human visual search. To prioritize search, GS assigns saliency to locations in the visual field. Saliency is a linear combination of activations from retinotopic maps representing primitive visual features. GS includes heuristics for setting the gain coefficient associated with each map. Variants of GS have formalized the notion of optimization as a principle of attentional control (e.g., Baldwin & Mozer, 2006; Cave, 1999; Navalpakkam & Itti, 2006; Rao et al., 2002), but every GS-like model must be 'dumbed down' to match human data, e.g., by corrupting the saliency map with noise and by imposing arbitrary restrictions on gain modulation. We propose a principled probabilistic formulation of GS, called Experience-Guided Search (EGS), based on a generative model of the environment that makes three claims: (1) Feature detectors produce Poisson spike trains whose rates are conditioned on feature type and whether the feature belongs to a target or distractor; (2) the environment and/or task is nonstationary and can change over a sequence of trials; and (3) a prior specifies that features are more likely to be present for target than for distractors. Through experience, EGS infers latent environment variables that determine the gains for guiding search. Control is thus cast as probabilistic inference, not optimization. We show that EGS can replicate a range of human data from visual search, including data that GS does not address.

## 1  Introduction

Human visual activity often involves search. We search for our keys on a cluttered desk, a face in a crowd, an exit sign on the highway, a key paragraph in a paper, our favorite brand of cereal at the supermarket, etc. The flexibility of the human visual system stems from the endogenous (or internal) control of attention, which allows for processing resources to be directed to task-relevant regions and objects in the visual field. How is attention directed based on an individual's goals? To what sort of features of the visual environment can attention be directed? These two questions are central to the study of how humans explore their environment.

Visual search has traditionally been studied in the laboratory using cluttered stimulus displays containing artificial objects. The objects are defined by a set of *primitive visual features*, such as color, shape, and size. For example, an experimental task might be to search for a red vertical line segment—the *target*—among green verticals and red horizontals—the *distractors*. Performance is typically evaluated as the response latency to detect the presence or absence of a target with high accuracy. The efficiency of visual search is often characterized by the search *slope*—the increase

Figure 1: The architecture of Guided Search

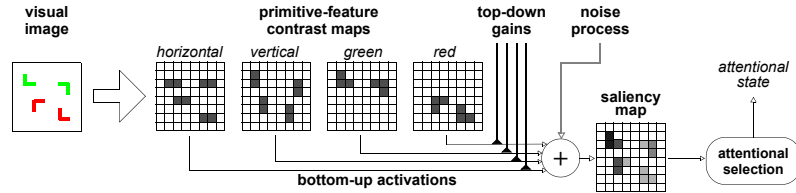

in response latency with each additional distractor in the display. An inefficient search can often require an additional 25–35 ms/item (or more, if eye movements are required).

Many computational models of visual search have been proposed to explain data from the burgeoning experimental literature (e.g., Baldwin & Mozer, 2006; Cave, 1999; Itti & Koch, 2001; Mozer, 1991; Navalpakkam & Itti, 2006; Sandon, 1990; Wolfe, 1994). Despite differences in their details, they share central assumptions, perhaps most plainly emphasized by Wolfe's (1994) Guided Search 2.0 (GS) model. We describe the central assumptions of GS, taking some liberty in ignoring details and complications of GS that obfuscate the similarities within this class of models and that are not essential for the purpose of this paper.[1]

As depicted Figure 1, GS posits that primitive visual features are detected across the retina in parallel along dimensions such as color and orientation, yielding a set of *feature activity maps*. Feature activations are scalars in $[0, 1]$. The feature maps represent each dimension via a coarse coding. That is, the maps for a particular dimension are highly overlapping and broadly tuned. For example, color might be represented by maps tuned to red, green, blue, and yellow; orientation might be represented by maps tuned to left, right, steep, and shallow-sloped edges. The feature activity maps are passed through a differencing mechanism that enhances local contrast and texture discontinuities, yielding a *bottom-up activation*.

The bottom-up activations from all feature maps are combined to form a *saliency map* in which activation at a location indicates the priority of that location for the task at hand. Attention is directed to locations in order from most salient to least, and the object at each location is identified. GS supposes that response latency is linear in the number of locations that need to be searched before a target is found. (The model includes rules for terminating search if no target is found after a reasonable amount of effort.)

Consider the task of searching for a red vertical bar among green vertical bars and red horizontal bars. Ideally, attention should be drawn to red and vertical bars, not to green or horizontal bar. To allow for guidance of attention, GS posits that a weight or *top-down gain* is associated with each feature map, and the contribution of given feature map to the saliency map is scaled by the gain. It is the responsibility of *control processes* to determining gains that emphasize task-relevant features.

Although gain modulation is a sensible means of implementing goal-directed action, it yields behavior than is more efficient than people appear to be. To elaborate, consider again the task of searching for a red vertical. If the gains on the red and vertical maps are set to 1, and the gains on green and horizontal maps are set to 0, then a target (red vertical) will have two units of activation in the saliency map, whereas each distractor (red horizontal or green vertical) will have only one unit of activation. Because the target is the most salient item and GS assumes that response time is monotonically related to the saliency ranking of the target, the target should be located quickly, in a time independent of the number of distractors. In contrast, human response times increase linearly with the number of distractors in conjunction search.

To reduce search efficiency, GS assumes noise corruption of the saliency map. In the case of GS, the signal-to-noise ratio is roughly 2:1. Baldwin and Mozer (2006) also require noise corruption for the same reason, although the corruption is to the low-level feature representation not the saliency map. Although Navalpakkam and Itti (2006) do not explicitly introduce noise in their model, they do so implicitly via a selection rule that the probability of attending an item is proportional to its saliency.

To further reduce search efficiency, GS includes a complex set of rules that limit gain control. For example, gain modulation is allowed for only one feature map per dimension. Other attentional models

place similar, somewhat arbitrary limitations on gain modulation. Baldwin and Mozer (2006) impose the restriction $\sum_i |g_i - 1| < c$, where $g_i$ is the gain of feature map $i$ and $c$ is a constant. Navalpakkam and Itti (2006) prefer the constraints $\sum_i g_i = c$ and $g_i > 0$.

Finally, we mention one other key property the various models have in common. Gain tuning is cast as an optimization problem: the goal of the model is to adjust the gains so as to maximize the target saliency relative to distractor saliency for the task at hand. Baldwin and Mozer (2006) define the criterion in terms of the target saliency ranking. Navalpakkam and Itti (2006) use the expected target to distractor saliency ratio. Wolfe (1994) sets gains according to rules that he describes as performing optimization.

## 2  Experience-Guided Search

The model we introduce in this paper makes three contributions over the class of Guided Search models previously proposed. (1) GS uses noise or nondeterminism to match human data. In reality, noise and nondeterminism serve to degrade the model's performance over what it could otherwise be. In contrast, all components of our model are justified on computational grounds, leading to a more elegant, principled account. (2) GS imposes arbitrary limitations on gain modulation that also result in the model performing worse than it otherwise could. Although limitations on gain modulation might be neurobiologically rationalized, a more elegant account would characterize these limitations in terms of trade offs: constraints on gain modulation may limit performance, but they yield certain benefits. Our model offers such a trade-off account. (3) In GS, attentional control is achieved by tuning gains to optimize performance. In contrast, our model is designed to infer the structure of its environment through experience, and gain modulation is a byproduct of this inference. Consequently, our model has no distinct control mechanism, leading to a novel perspective on executive control processes in the brain.

Our approach begins with the premise that attention is fundamentally task based: a location in the visual field is salient if a target is likely at that location. We define saliency as the *target probability*, $P(T_x = 1 | \mathbf{F}_x)$, where $\mathbf{F}_x$ is the local feature activity vector at retinal location $x$ and $T_x$ is a binary random variable indicating if location $x$ contains a target. Torralba et al. (2006) and Zhang and Cottrell (submitted) have also suggested that saliency should reflect target probability, although they propose approaches to computing the target probability very different from ours. Our approach is to compute the target probability using statistics obtained from recent experience performing the task. Consequently, we refer to our model as *experience-guided search* or *EGS*.

To expand $P(T_x | \mathbf{F}_x)$, we make the naive-Bayes assumption that the feature activities are independent of one another, yielding

$$P(T_x | \mathbf{F}_x, \boldsymbol{\rho}) = P(T_x) \prod_i P(F_{xi} | T_x, \boldsymbol{\rho}) / \sum_{t=0}^{1} P(T_x = t) \prod_i P(F_{xi} | T_x = t, \boldsymbol{\rho}), \quad (1)$$

where $\boldsymbol{\rho}$ is a vector of parameters that characterize the current stimulus environment in the current task, and $F_{xi}$ encodes the activity of feature $i$. Consider $F_{xi}$ to be a rate-coded representation of a neural spike train. Specifically, $F_{xi}$ denotes the *count* of the number of spikes that occurred in a window of $n$ time intervals, where at most one spike can occur in each interval.

We propose a generative model of the environment in which $F_{xi}$ is a binomial random variable, $F_{xi} | \{T_x = t, \boldsymbol{\rho}\} \sim \text{Binomial}(\rho_{it}, n)$, where a spike rate $\rho_{it}$ is associated with feature $i$ for target ($t = 1$) and nontarget ($t = 0$) items. As $n$ becomes large—i.e., the spike count is obtained over a larger period of time—the binomial is well approximated by a Gaussian: $F_{xi} | \{T_x = t, \boldsymbol{\rho}\} \sim \mathcal{N}(n\rho_{it}, n\rho_{it}(1 - \rho_{it}))$. Using the Gaussian approximation, Equation 1 can be rewritten in the form of a logistic function: $P(T_x | \mathbf{F}_x, \boldsymbol{\rho}) = 1/(1 + e^{-(r_x + \frac{n}{2} s_x)})$, where

$$r_x = \ln\left[\frac{P(T_x = 1)}{P(T_x = 0)}\right] - \frac{1}{2} \sum_i \ln\left[\frac{\rho_{i1}(1 - \rho_{i1})}{\rho_{i0}(1 - \rho_{i0})}\right] \text{ and } s_x = \sum_i \sum_{t=0}^{1} \frac{1 - 2t}{\rho_{it}(1 - \rho_{it})} (\tilde{f}_{xi} - \rho_{it})^2 \quad (2)$$

and $\tilde{f}_{xi} = f_{xi}/n$ denotes the observed spike *rate* for a feature detector.

Because of the logistic relationship, $P(T_x | \mathbf{F}_x, \boldsymbol{\rho})$ is monotonic in $r_x + \frac{n}{2} s_x$. Consequently, if attentional priority is given to locations in order of their target probability, $P(T_x | \mathbf{F}_x, \boldsymbol{\rho})$, then it is

equivalent to rank using $r_x + \frac{n}{2}s_x$. Further, if we assume that the target is equally likely in any location, then $r_x$ is constant across locations, and $s_x$ can substitute for $P(T_x|\boldsymbol{F}_x, \boldsymbol{\rho})$ as an equivalent measure of saliency.

This saliency measure, $s_x$, makes intuitive sense. Saliency at a location increases if feature $i$'s activity is distant from the mean activity observed in the past for a distractor ($\rho_{i0}$) and decreases if feature $i$'s activity is distant from the mean activity observed in the past for a target ($\rho_{i1}$). These saliency increases (decreases) are scaled by the variance of the distractor (target) activities, such that high-variance features have less impact on saliency.

Expanding the numerator terms in the definition of $s_x$ (Equation 2), we observe that $s_x$ can be written as a linear combination of terms involving the feature activities, $\tilde{f}_{xi}$, and the squared activities, $\tilde{f}_{xi}^2$ (along with a constant term that can be ignored for ranking by saliency). The saliency measure $s_x$ in EGS is thus quite similar to the saliency measure in GS, $s_x^{\mathrm{GS}} = \sum_i g_i \tilde{f}_{xi}$. The differences are: first, EGS incorporates quadratic terms, and second, gain coefficients of EGS are not free parameters but are derived from statistics of targets and distractors in the current task and stimulus environment. In this fact lies the virtue of EGS relative to GS: The control parameters are obtained not by optimization, but are derived directly from statistics of the environment.

## 2.1 Uncertainty in the Environment Statistics

The model parameters, $\boldsymbol{\rho}$, could be maximum likelihood estimates obtained by observing target and distractor activations over a series of trials. That is, suppose that each item in the display is identified as a target or distractor. The set of activations of feature $i$ at all locations containing a target could be used to estimate $\rho_{i1}$, and likewise with locations containing a distractor to estimate $\rho_{i0}$. Alternatively, one could adopt a Bayesian approach and treat $\rho_{it}$ as a random variable, whose uncertainty is reduced by the evidence obtained on each trial. Because feature spike rates lie in $[0, 1]$, we define $\rho_{it}$ as a beta random variable, $\rho_{it} \sim \mathrm{Beta}(\alpha_{it}, \beta_{it})$.

This Bayesian approach also allows us to specify priors over $\rho_{it}$ in terms of imaginary counts, $\alpha_{it}^0$ and $\beta_{it}^0$. For example, in the absence of any task experience, a conservative assumption is that all feature activations are predictive of a target, i.e., $\rho_{i1}$ should be drawn from a distribution biased toward 1, and $\rho_{i0}$ should be drawn from a distribution biased toward 0.

To compute the target probabilities, we must marginalize over $\boldsymbol{\rho}$, i.e., $P(T_x|\boldsymbol{F}_x) = \int_{\boldsymbol{\rho}} P(T_x|\boldsymbol{F}_x, \boldsymbol{\rho})p(\boldsymbol{\rho})d\boldsymbol{\rho}$. Unfortunately, this integral is impossible to evaluate analytically. We instead compute the expectation of $s_x$ over $\boldsymbol{\rho}$, $\bar{s}_x \equiv E_{\boldsymbol{\rho}}(s_x)$, which has the solution

$$\bar{s}_x = \sum_i \sum_{t=0}^{1}(1-2t)\left[\frac{(\alpha_{it}+\beta_{it}-1)(\alpha_{it}+\beta_{it}-2)}{(\alpha_{it}-1)(\beta_{it}-1)}\tilde{f}_{xi}^2 - \frac{2(\alpha_{it}+\beta_{it}-1)}{\beta_{it}-1}\tilde{f}_{xi} + \frac{\alpha_{it}}{\beta_{it}-1}\right] \quad (3)$$

Note that, like the expression for $s_x$, $\bar{s}_x$ is a weighted sum of linear and quadratic feature-activity terms. When $\alpha_{it}$ and $\beta_{it}$ are large, the distribution of $\rho_{it}$ is sharply peaked, and $\bar{s}_x$ approaches $s_x$ with $\rho_{it} = \alpha_{it}/(\alpha_{it}+\beta_{it})$. When this condition is satisfied, ranking by $\bar{s}_x$ is equivalent to ranking by $P(T_x|\boldsymbol{F}_x)$. Although the equivalence is not guaranteed for smaller $\alpha_{it}$ and $\beta_{it}$, we have found the equivalence to hold in empirical tests. Indeed, in our simulations, we find that defining saliency as either $s_x$ or $\bar{s}_x$ yields similar results, reinforcing the robustness of our approach.

## 2.2 Modeling the Stimulus Environment

The parameter vectors $\boldsymbol{\alpha}$ and $\boldsymbol{\beta}$ maintain a model of the stimulus environment in the context of the current task. Following each trial, these parameters must be updated to reflect the statistics of the trial. We assume that following a trial, each item in the display has been identified as either a target or distractor. (All other adaptive attention models such as GS make this assumption.)

Consider a location $x$ that has been labeled as type $t$ (1 for target, 0 for distractor), and some feature $i$ at that location, $F_{xi}$. We earlier characterized $F_{xi}$ as a binomial random variable reflecting a spike count; that is, during $n$ time intervals, $f_{xi}$ spikes are observed. Each time interval provides evidence as to the value $\rho_{it}$. Given prior distribution $\rho_{it} \sim \mathrm{Beta}(\alpha_{it}, \beta_{it})$, the posterior is $\rho_{it}|F_{xi} \sim \mathrm{Beta}(\alpha_{it}+f_{xi}, \beta_{it}+n-f_{xi})$. However, to limit the evidence provided from each item, we scale it

by a factor of $n$. When all locations are considered, the resulting posterior is:

$$\rho_{it}|\boldsymbol{F}_i \sim \text{Beta}\left(\alpha_{it} + \sum_{x\in\chi_t}\tilde{f}_{xi}, \quad \beta_{it} + \sum_{x\in\chi_t} 1 - \tilde{f}_{xi}\right) \tag{4}$$

where $\boldsymbol{F}_i$ is feature map $i$ and $\chi_t$ is the set of locations containing elements of type $t$.

With the approach we've described, evidence concerning the value of $\rho_{it}$ accumulates over a sequence of trials. However, if an environment is nonstationary, this build up of evidence is not adaptive. We thus consider a switching model of the environment that specifies with probability $\lambda$, the environment changes and all evidence should be discarded. The consequence of this assumption is that the posterior distribution is a mixture of Equation 4 and the prior distribution, $\text{Beta}(\alpha_{it}^0, \beta_{it}^0)$.

Modeling the mixture distribution is problematic because the number of mixture components grows linearly with the number of trials. We could approximate the mixture distribution by the beta distribution that best approximates the mixture, in the sense of Kullback-Leibler divergence. However, we chose to adopt a simpler, more intuitive solution: to interpolate between the two distributions. The update rule we use is therefore

$$\rho_{it}|\boldsymbol{F}_i \sim \text{Beta}\left(\lambda\alpha_{it}^0 + (1-\lambda)\left[\alpha_{it} + \sum_{x\in\chi_t}\tilde{f}_{xi}\right], \quad \lambda\beta_{it}^0 + (1-\lambda)\left[\beta_{it} + \sum_{x\in\chi_t} 1 - \tilde{f}_{xi}\right]\right). \tag{5}$$

## 3  Simulation Methodology

We present a step-by-step description of how the model runs to simulate experimental subjects performing a visual search task. We start by generating a sequence of experimental trials with the properties studied in an experiment. The model is initialized with $\alpha_{it} = \alpha_{it}^0$ and $\beta_{it} = \beta_{it}^0$. On each simulation trial, the following sequence occurs. (1) Feature extraction is performed on the display to obtain firing rates, $\tilde{f}_{xi}$ for each location $x$ and feature type $i$. (2) Saliency, $\bar{s}_x$, is computed for each location according to Equation 3. (3) The saliency *rank* of each location is assessed, and the number of locations that need to be searched in order to identify the target is assumed to be equal to the target rank. Response time should then be linear in target rank. (4) Following each trial, target and distractor statistics, $\alpha_{it}$ and $\beta_{it}$, are updated according to Equation 5.

EGS has potentially many free parameters: $\{\alpha_{it}^0\}$ and $\{\beta_{i1}^0\}$, and $\lambda$. However, with no reason to believe that one feature behaves differently than another, we assign all the features the same priors. Further, we impose symmetry such that $\alpha_{i0}^0 = \beta_{j1}^0 = \nu$ and $\alpha_{i1}^0 = \beta_{j0}^0 = \mu$ for all $i$ and $j$, reducing the total number of free parameters to three.

Because we are focused on the issue of attentional control, we wanted to sidestep other issues, such as feature extraction. Consequently, EGS uses the front-end preprocessing of GS. GS takes as input an $8 \times 8$ array of locations, each of which can contain a single colored bar. As described earlier, GS analyzes the input via four broadly tuned features for color, and four for orientation. After a local contrast-enhancement operator, GS yields activation values in $[0, 1]$ at each of $8 \times 8$ locations for each of eight feature dimensions. We treat the activation produced by GS for feature $i$ at location $x$ as the firing rate $\tilde{f}_{xi}$ needed to simulate EGS. Like GS, the response time of EGS is linear in the target ranking. A scaling factor is required to convert rank to response time; we chose 25 msec/item, which is a fourth free parameter of GS.

## 4  Results

We simulated EGS on a series of tasks that Wolfe (1994) used to evaluate GS. Because GS is limited to processing displays containing colored, oriented lines, some of the tasks constructed by Wolfe did not have an exact correspondence in the experimental literature. Rather, Wolfe, the leading expert in visual search, identified key findings that he wanted GS to replicate. Because EGS shares front-end processing with GS, EGS is limited to the same set of tasks as GS. Consequently, we present a comparison of GS and EGS.

We began by replicating Wolfe's results on GS. This replication was nontrivial, because GS contains many parameters, rules, and special cases, and published descriptions of GS do not provide a crisp

algorithmic description of the model. To implement EGS, we simply removed much of the complexity of GS—including the distinction between bottom-up and top-down weights, heuristics for setting the weights, and the injection of high-amplitude noise into the saliency map—and replaced it with Equations 3 and 5.

Each simulation begins with a sequence of 100 practice trials, followed by a sequence of 1000 trials for each blocked condition. Displays on each trial are generated according to the constraints of the task with random variation with respect to unconstrained aspects of the task (e.g., locations of display elements, distractor identities, etc.). In typical search tasks, the participant is asked to detect the presence or absence of a target. We omit results for target-absent trials, since GS and EGS make identical predictions for these trials.

The qualitative performance of EGS does not depend on its free parameters when two conditions are met: $\lambda > 0$ and $\mu > \nu$. The latter condition yields $E[\rho_{i1}] > E[\rho_{i0}]$ for all $i$, and corresponds to the bias that features are more likely to be present for a target than for a distractor. This bias is rational in order to prevent cognition from suppressing information that could potentially be critical to behavior. All simulation results reported here used $\lambda = 0.3$, $\mu = 25$, and $\nu = 10$.

Figure 2 shows simulation results on six sets of tasks, labeled A–F. The first and third columns (thin lines) are data from our replication of GS; the second and fourth columns (thick lines) are data from our implementation of EGS. The key feature to note is that results from EGS are qualitatively and quantitatively similar to results from GS. As should become clear when we explain the individual tasks, EGS probably produces a better qualitative fit to the human data. (Unfortunately, it is not feasible to place the human data side-by-side with the simulation results. Although the six sets of tasks were chosen by Wolfe to represent key experiments in the literature, most are abstractions of the original experimental tasks because the retina of GS—and its descendent EGS—is greatly simplified and cannot accommodate the stimulus arrays used in human studies. Thus, Wolfe never intended to quantitatively model specific experimental studies.)

We briefly describe the six tasks. The first four involve displays of a homogeneous color, and search for a target orientation among distractors of different orientations. *Task A* explores search for a vertical (defined as $0°$) target among homogeneous distractors of a different orientation. The graph plots the slope of the line relating display size to response latency, as a function of the distractor orientation. Search slopes become more efficient as the target-distractor similarity decreases. *Task B* explores search for a target among two types of distractors as a function of display size. The distractors are $100°$ apart, and the target is $40°$ and $60°$ from the distractors, but in one case the target differs from the distractors in that it is the only nearly vertical item, allowing pop out via the vertical feature detector. Note that pop out is not wired into EGS, but emerges because EGS identifies vertical-feature activity as a reliable predictor of the target. *Task C* examines search efficiency for a target among heterogeneous distractors, for two target orientations and two degrees of target-distractor similarity. Search is more efficient when the target and distractors are dissimilar. (EGS obtains results better matched to the human data than GS.) *Task D* explores an asymmetry in search: it is more efficient to find a tilted bar among verticals than a vertical among tilted. This effect arises from the same mechanism that yielded efficient search in task B: a unique feature is highly activated when the target is tilted but not when it is vertical. And search is better guided to features that are present than to features that are absent in EGS, due to the $\boldsymbol{\rho}$ priors. *Task E* involves conjunction search. The target is a red vertical among green vertical and red tilted distractors. The red item's tilt can be either $90°$ (i.e., horizontal) or $40°$. Both distractor environments yield inefficient search, but—consistent with human data—conjunction searches can vary in their relative difficulty.

*Task F* examines search efficiency for a red vertical among red $60°$ and yellow vertical distractors, as a function of the ratio of the two distractor types. The result shows that search can be guided: response times become faster as either the target color or target orientation becomes sparse, because a relatively unique feature serves as a reliable cue to the target. Figure 3a depicts how EGS adapts differently for the extreme conditions in which the distractors are mostly vertical (dark bars) or mostly red (light bars). The bars represent $E[\rho_{i0}]$; the lower the value, the more a feature is viewed as reliably discriminating targets and distractors. ($E[\rho_{i1}]$ is independent of the experimental condition.) When distractors are mostly vertical, the red feature is a better cue, and vice versa. The standard explanation for this phenomenon in the psychological literature is that subjects operate in two stages, first filtering out based on the more discriminative feature, and then serially searching the remaining

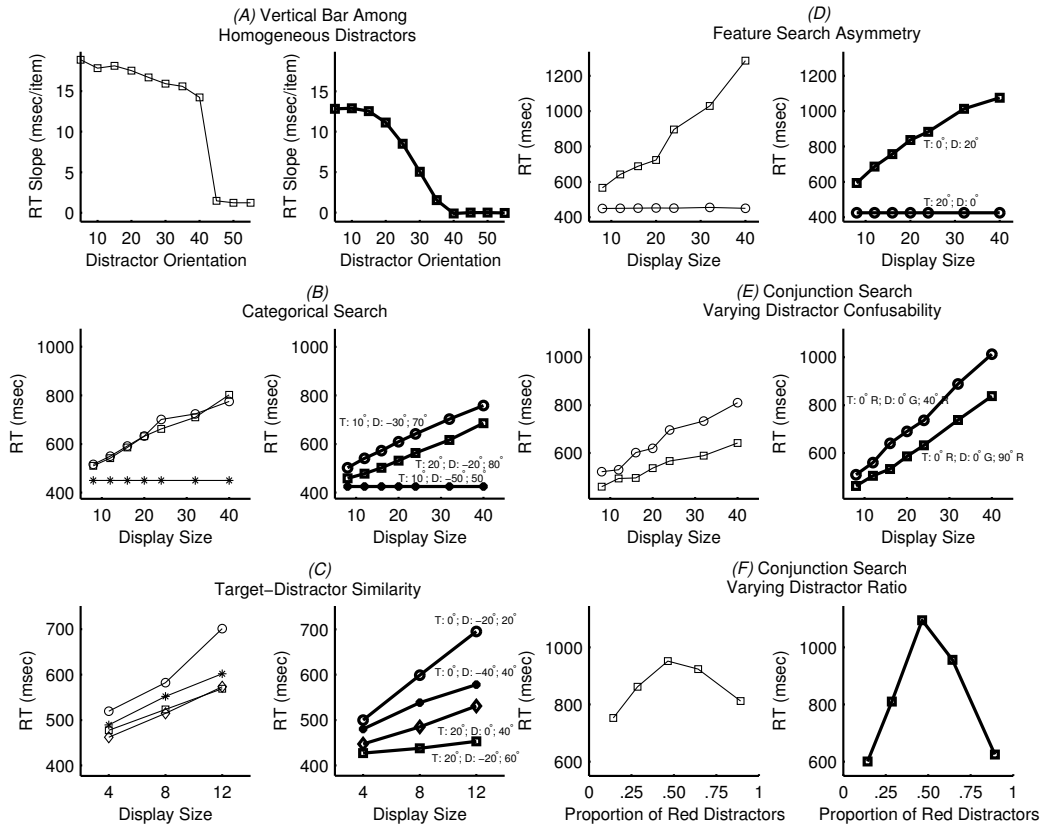

Figure 2: Simulation results on six sets of tasks, labeled A–F, for GS (thin lines, 1st and 3d columns) and EGS (thick lines, 2nd and 4th columns). Simulation details are explained in the text.

items. EGS provides a single-stage account that does not need to invoke specialized mechanisms for adaptation to the environment, because all attentional control is adaptation of this sort.

To summarize, EGS predicts the key factors in visual search that determine search efficiency. Most efficient search is for a target defined by the presence of a single categorical feature among homogeneous distractors that do not share the categorical feature. Least efficient search is for target and distractors that share features (e.g., T among L's, or red verticals among red horizontals and green verticals) and/or when distractors are heterogeneous.

Wolfe, Cave, & Franzel (1989) conducted an experiment to demonstrate that people can benefit from guidance. This experiment, which oddly has never been modeled by GS, involves search for a conjunction target defined by a triple of features, e.g., a big red vertical bar. The target might be presented among heterogeneous distractors that share two features with it, such as a big red horizontal bar, or distractors that share only one feature with it, such as a small green vertical bar. Performance in these two conditions, denoted T3-D2 and T3-D1, respectively, is compared to performance in a standard conjunction search task, denoted T2-D1, involving targets defined by two features and sharing one feature with each distractor. Wolfe et al. reasoned that if search can be guided, saliency at a location should be proportional to the number of target-relevant features at that location, and the ratio of target to distractor salience should be $x/y$ in condition T$x$-D$y$. Because $x > y$, the target is always more salient than any distractor, but GS assumes less efficient search due to noise corruption of the saliency map, thereby predicting search slopes that are inversely related to $x/y$. The human data show exactly this pattern, producing almost flat search slopes for T3-D1. EGS replicates the human data (Figure 3b) without employing GS's arbitrary assumption that prioritization is corrupted by noise. Instead, $x/y$ reflects the amount of evidence available on each trial about features that discriminate targets from distractors. Essentially, EGS suggests that $x/y$ determines the availability of discriminative statistics in the environment. Thus, the limitation is on learning, not on performance.

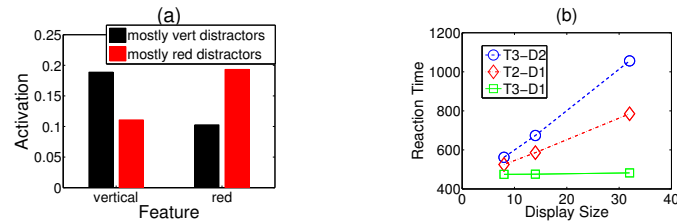

Figure 3: (a) Values of $E[\rho_{i0}]$ in task F. (b) EGS performance on the triple-conjunction task of Wolfe, Cave, & Franzel (1989)

## 5   Discussion

We presented a model, EGS, that guides visual search via statistics collected over the course of experience in a task environment. The primary contributions of EGS are as follows. First, EGS is a significantly more elegant and parsimonious theory than its predecessors. In contrast to EGS, GS is a complex model under the hood with many free parameters and heuristic assumptions. We and other groups have spent many months reverse engineering GS to determine how exactly it works, because published descriptions do not have the specificity of an algorithm. Second, to explain human data, GS and its ancestors are "retarded" by injecting noise or arbitrarily limiting gains. Although it may ultimately be determined that the brain suffers from these conditions, one would prefer theories that cast performance of the brain as ideal or rational. EGS achieves this objective via explicit assumptions about the generative model of the environment embodied by cognition. In particular, the dumbing-down of GS and its variants is replaced in EGS by the claim that environments are nonstationary. If the environment can change from one trial to the next, the cognitive system does well not to turn up gains on one feature dimension at the expense of other feature dimensions. The result is a sensible trade off: attentional control can be rapidly tuned as the task or environment changes, but this flexibility restricts EGS's search efficiency when the task and environment remain constant. Third, EGS suggests a novel perspective on attentional control, and executive control more generally. All other modern perspectives we are aware of treat control as *optimization*, whereas in EGS, control arises directly from statistical *inference* on the task environment. Our current research is exploring the implications of this intriguing perspective.

**Acknowledgments**

This research was supported by NSF BCS 0339103 and NSF CSE-SMA 0509521. Support for the second author comes from an NSF Graduate Fellowship.

**References**

Baldwin, D., & Mozer, M. C. (2006). Controlling attention with noise: The cue-combination model of visual search. In R. Sun & N. Miyake (Eds.), *Proc. of the 28th Ann. Conf. of the Cog. Sci. Society* (pp. 42-47). Hillsdale, NJ: Erlbaum.

Cave, K. R. (1999). The FeatureGate model of visual selection. *Psychol. Res.*, *62*, 182–194.

Itti, L., & Koch, C. (2001). Computational modeling of visual attention. *Nature Rev. Neurosci.*, *2*, 194–203.

Mozer, M. C. (1991). The perception of multiple objects: A connectionist approach. Cambridge, MA: MIT.

Navalpakkam, V., & Itti, L. (2006). Optimal cue selection strategy. In *Advances in Neural Information Processing Systems Vol. 19* (pp. 1-8). Cambridge, MA: MIT Press.

Rao, R., Zelinsky, G., Hayhoe, M., & Ballard, D. (2002). Eye movements in iconic visual search. *Vis. Res.*, *42*, 1447–1463.

Sandon, P. A. (1990). Simulating visual attention. *Journal of Cog. Neuro.*, *2*, 213–231. Sandon, 1990

Torralba, A., Oliva, A., Castelhano, M.S., & Henderson, J. M. (2006). Contextual guidance of eye movements and attention in real-world scenes: The role of global features on objects search. *Psych. Rev.*, *113*, 766–786.

Wolfe, J. M., Cave, K. R., & Franzel, S. L. (1989). Guided search: An alternative to the feature integration model for visual search. *Jnl. Exp. Psych.: Hum. Percep. & Perform.*, *15*, 419–433.

Wolfe, J. M. (1994). Guided Search 2.0: A revised model of visual search. *Psych. Bull. & Rev.*, *1*, 202–238.

Wolfe, J. M. (2007). Guided Search 4.0: Current progress with a model of visual search. In. W. Gray (Ed.), *Integrated Models of Cognitive Systems*. NY: Oxford.

Zhang, L., & Cottrell, G. W. (submitted). Probabilistic search: A new theory of visual search. Submitted for publication.

## Footnotes

[1]Although Guided Search has undergone refinement (Wolfe, 2007), the key claims summarized here are unchanged. Recent extensions to GS consider eye movements and acuity changes with retinal eccentricity.
